# On the Effect of Analog Noise in Discrete-Time Analog Computations

**Wolfgang Maass**
Institute for Theoretical Computer Science
Technische Universität Graz*

**Pekka Orponen**
Department of Mathematics
University of Jyväskylä[†]

## Abstract

We introduce a model for noise-robust analog computations with discrete time that is flexible enough to cover the most important concrete cases, such as computations in noisy analog neural nets and networks of noisy spiking neurons. We show that the presence of arbitrarily small amounts of analog noise reduces the power of analog computational models to that of finite automata, and we also prove a new type of upper bound for the VC-dimension of computational models with analog noise.

## 1  Introduction

Analog noise is a serious issue in practical analog computation. However there exists no formal model for reliable computations by noisy analog systems which allows us to address this issue in an adequate manner. The investigation of noise-tolerant *digital* computations in the presence of stochastic failures of gates or wires had been initiated by [von Neumann, 1956]. We refer to [Cowan, 1966] and [Pippenger, 1989] for a small sample of the numerous results that have been achieved in this direction. In all these articles one considers computations which produce a correct output not with perfect reliability, but with probability $\geq \frac{1}{2} + \rho$ (for some parameter $\rho \in (0, \frac{1}{2}]$). The same framework (with stochastic failures of gates or wires) has been applied to analog neural nets in [Siegelmann, 1994].

The abovementioned approaches are insufficient for the investigation of noise in analog computations, because in analog computations one has to be concerned not only with occasional total failures of gates or wires, but also with "imprecision", i.e. with omnipresent smaller (and occasionally larger) perturbations of analog outputs

of internal computational units. These perturbations may for example be given by Gaussian distributions. Therefore we introduce and investigate in this article a notion of noise-robust computation by noisy analog systems where we assume that the values of intermediate analog values are moved according to some quite arbitrary probability distribution. We consider – as in the traditional framework for noisy *digital* computations – arbitrary computations whose output is correct with some given probability $\geq \frac{1}{2} + \rho$ (for $\rho \in (0, \frac{1}{2}]$) . We will restrict our attention to analog computation with *digital* output. Since we impose no restriction (such as continuity) on the type of operations that can be performed by computational units in an analog computational system, an output unit of such system can convert an analog value into a binary output via "thresholding".

Our model and our Theorem 3.1 are somewhat related to the analysis of probabilistic finite automata in [Rabin, 1963]. However there the finiteness of the state space simplifies the setup considerably. [Casey, 1996] addresses the special case of analog computations on recurrent neural nets (for those types of analog noise that can move an internal state at most over a distance $\varepsilon$) whose digital output is *perfectly reliable* (i.e. $\rho = 1/2$ in the preceding notation).[1]

The restriction to perfect reliability in [Casey, 1996] has immediate consequences for the types of analog noise processes that can be considered, and for the types of mathematical arguments that are needed for their investigation. In a computational model with perfect reliability of the output it cannot happen that an intermediate state $\underline{s}$ occurs at some step $t$ both in a computation for an input $\underline{x}$ that leads to output "0" , and at step $t$ in a computation for the same input "$\underline{x}$" that leads to output "1" . Hence an analysis of perfectly reliable computations can focus on *partitions* of intermediate states $\underline{s}$ according to the computations and the computation steps where they may occur.

Apparently many important concrete cases of noisy analog computations require a different type of analysis. Consider for example the special case of a sigmoidal neural net (with thresholding at the output), where for each input the output of an internal noisy sigmoidal gate is distributed according to some Gaussian distribution (perhaps restricted to the range of all possible output values which this sigmoidal gate can actually produce). In this case an intermediate state $\underline{s}$ of the computational system is a vector of values which have been produced by these Gaussian distributions. Obviously each such intermediate state $\underline{s}$ can occur at any fixed step $t$ in *any* computation (in particular in computations with different network output for the same network input). Hence perfect reliability of the network output is unattainable in this case. For an investigation of the actual computational power of a sigmoidal neural net with Gaussian noise one has to drop the requirement of perfect reliability of the output, and one has to analyze how *probable* it is that a particular network output is given, and how *probable* it is that a certain intermediate state is assumed. Hence one has to analyze for each network input and each step $t$ the different

*probability distributions* over intermediate states $\underline{s}$ that are induced by computations of the noisy analog computational system. In fact, one may view the set of these probability distributions over intermediate states $\underline{s}$ as a generalized set of "states" of a noisy analog computational system. In general the mathematical structure of this generalized set of "states" is substantially more complex than that of the original set of intermediate states $\underline{s}$. We have introduced in [Maass, Orponen, 1996] some basic methods for analyzing this generalized set of "states", and the proofs of the main results in this article rely on this analysis.

The preceding remarks may illustrate that if one drops the assumption of perfect reliability of the output, it becomes more difficult to prove *upper* bounds for the power of noisy analog computations. We prove such upper bounds even for the case of *stochastic dependencies* among noises for different internal units, and for the case of *nonlinear dependencies* of the noise on the current internal state. Our results also cover noisy computations in hybrid analog/digital computational models, such as for example a neural net combined with a binary register, or a network of noisy spiking neurons where a neuron may temporarily assume the discrete state "not-firing". Obviously it becomes quite difficult to analyze the computational effect of such complex (but practically occuring) types of noise without a rigorous mathematical framework. We introduce in section 2 a mathematical framework that is general enough to subsume all these cases. The traditional case of noisy *digital* computations is captured as a special case of our definition.

One goal of our investigation of the effect of analog noise is to find out *which* features of analog noise have the most detrimental effect on the computational power of an analog computational system. This turns out to be a nontrivial question.[2] As a first step towards characterizing those aspects and parameters of analog noise that have a strong impact on the computational power of a noisy analog system, the proof of Theorem 3.1 (see [Maass, Orponen, 1996]) provides an explicit bound on the number of states of any finite automaton that can be implemented by an analog computational system with a given type of analog noise. It is quite surprising to see on which specific parameters of the analog noise the bound depends. Similarly the proofs of Theorem 3.4 and Theorem 3.5 provide explicit (although very large) upper bounds for the VC-dimension of noisy analog neural nets with batch input, which depend on specific parameters of the analog noise.

## 2    Preliminaries: Definitions and Examples

An *analog discrete-time computational system* (briefly: *computational system*) $M$ is defined in a general way as a 5-tuple $\langle \Omega, p^0, F, \Sigma, s \rangle$, where $\Omega$, the set of *states*, is a bounded subset of $\mathbf{R}^d$, $p^0 \in \Omega$ is a distinguished *initial state*, $F \subseteq \Omega$ is the set of *accepting states*, $\Sigma$ is the *input domain*, and $s : \Omega \times \Sigma \to \Omega$ is the *transition function*. To avoid unnecessary pathologies, we impose the conditions that $\Omega$ and $F$ are Borel subsets of $\mathbf{R}^d$, and for each $a \in \Sigma$, $s(p, a)$ is a measurable function of $p$. We also assume that $\Sigma$ contains a distinguished null value $\sqcup$, which may be used to pad the actual input to arbitrary length. The nonnull input domain is denoted by $\Sigma_0 = \Sigma - \{\sqcup\}$.

The intended noise-free dynamics of such a system $M$ is as follows. The system starts its computation in state $p^0$, and on each single computation step on input element $a \in \Sigma_0$ moves from its current state $p$ to its next state $s(p, a)$. After the actual input sequence has been exhausted, $M$ may still continue to make pure computation steps. Each pure computation step leads it from a state $p$ to the state $s(p, \sqcup)$. The system accepts its input if it enters a state in the class $F$ at some point after the input has finished.

For instance, the *recurrent analog neural net* model of [Siegelmann, Sontag, 1991] (also known as the *"Brain State in a Box"* model) is obtained from this general framework as follows. For a network $\mathcal{N}$ with $d$ neurons and activation values between $-1$ and $1$, the state space is $\Omega = [-1, 1]^d$. The input domain may be chosen as either $\Sigma = \mathbf{R}$ or $\Sigma = \{-1, 0, 1\}$ (for "online" input) or $\Sigma = \mathbf{R}^n$ (for "batch" input).

*Feedforward analog neural nets* may also be modeled in the same manner, except that in this case one may wish to select as the state set $\Omega := ([-1, 1] \cup \{dormant\})^d$, where *dormant* is a distinguished value not in $[-1, 1]$. This special value is used to indicate the state of a unit whose inputs have not all yet been available at the beginning of a given computation step (e.g. for units on the $l$-th layer of a net at computation steps $t < l$).

The completely different model of a *network of $m$ stochastic spiking neurons* (see e.g. [Maass, 1997]) is also a special case of our general framework.[3]

Let us then consider the *effect of noise in a computational system $M$*. Let $Z(p, B)$ be a function that for each state $p \in \Omega$ and Borel set $B \subseteq \Omega$ indicates the probability of noise moving state $p$ to some state in $B$. The function $Z$ is called the *noise process affecting $M$*, and it should satisfy the mild conditions of being a *stochastic kernel*, i.e., for each $p \in \Omega$, $Z(p, \cdot)$ should be a probability distribution, and for each Borel set $B$, $Z(\cdot, B)$ should be a measurable function.

We assume that there is some measure $\mu$ over $\Omega$ so that $Z(p, \cdot)$ is absolutely continuous with respect to $\mu$ for each $p \in \Omega$, i.e. $\mu(B) = 0$ implies $Z(p, B) = 0$ for every measurable $B \subseteq \Omega$. By the Radon–Nikodym theorem, $Z$ then possesses a *density kernel* with respect to $\mu$, i.e. there exists a function $z(\cdot, \cdot)$ such that for any state $p \in \Omega$ and Borel set $B \subseteq \Omega$, $Z(p, B) = \int_{q \in B} z(p, q) \, d\mu$.

We assume that this function $z(\cdot, \cdot)$ has values in $[0, \infty)$ and is measurable. (Actually, in view of our other conditions this can be assumed without loss of generality.)

The dynamics of a computational system $M$ affected by a noise process $Z$ is now defined as follows.[4] If the system starts in a state $p$, the distribution of states $q$ obtained after a single computation step on input $a \in \Sigma$ is given by the density kernel $\pi_a(p, q) = z(s(p, a), q)$. Note that as a composition of two measurable func-

tions, $\pi_a$ is again a measurable function. The long-term dynamics of the system is given by a Markov process, where the distribution $\pi_{xa}(p, q)$ of states after $|xa|$ computation steps with input $xa \in \Sigma^*$ starting in state $p$ is defined recursively by $\pi_{xa}(p, q) = \int_{r \in \Omega} \pi_x(p, r) \cdot \pi_a(r, q) \, d\mu$.

Let us denote by $\pi_x(q)$ the distribution $\pi_x(p^0, q)$, i.e. the distribution of states of $M$ after it has processed string $x$, starting from the initial state $p^0$. Let $\rho > 0$ be the required reliability level. In the most basic version the system $M$ accepts (rejects) some input $x \in \Sigma_0^*$ if $\int_F \pi_x(q) \, d\mu \geq \frac{1}{2} + \rho$ (respectively $\leq \frac{1}{2} - \rho$). In less trivial cases the system may also perform pure computation steps after it has read all of the input. Thus, we define more generally that the system $M$ *recognizes a set* $L \subseteq \Sigma_0^*$ *with reliability* $\rho$ if for any $x \in \Sigma_0^*$:

$$x \in L \quad \Leftrightarrow \quad \int_F \pi_{xu}(q) \, d\mu \geq \frac{1}{2} + \rho \text{ for some } u \in \{\sqcup\}^*$$

$$x \notin L \quad \Leftrightarrow \quad \int_F \pi_{xu}(q) \, d\mu \leq \frac{1}{2} - \rho \text{ for all } u \in \{\sqcup\}^*.$$

This covers also the case of batch input, where $|x| = 1$ and $\Sigma_0$ is typically quite large (e.g. $\Sigma_0 = \mathbf{R}^n$).

## 3   Results

The proofs of Theorems 3.1, 3.4, 3.5 require a mild continuity assumption for the density functions $z(r, \cdot)$, which is satisfied in all concrete cases that we have examined. We do *not* require any *global* continuity property over $\Omega$ for the density functions $z(r, \cdot)$ because there are important special cases (see [Maass, Orponen, 1996]), where the state space $\Omega$ is a disjoint union of subspaces $\Omega_1, \ldots, \Omega_k$ with different measures on each subspace. We only assume that for some arbitrary partition of $\Omega$ into Borel sets $\Omega_1, \ldots, \Omega_k$ the density functions $z(r, \cdot)$ are uniformly continuous over each $\Omega_j$, with moduli of continuity that can be bounded independently of $r$. In other words, we require that $z(\cdot, \cdot)$ satisfies the following condition:

We call a function $\pi(\cdot, \cdot)$ from $\Omega^2$ into $\mathbf{R}$ *piecewise uniformly continuous* if for every $\varepsilon > 0$ there is a $\delta > 0$ such that for every $r \in \Omega$, and for all $p, q \in \Omega_j$, $j = 1, \ldots, k$:

$$\| p - q \| \leq \delta \quad \text{implies} \quad |\pi(r, p) - \pi(r, q)| \leq \varepsilon. \tag{1}$$

If $z(\cdot, \cdot)$ satisfies this condition, we say that the resulting noise process $Z$ is *piecewise uniformly continuous*.

**Theorem 3.1** *Let $L \subseteq \Sigma_0^*$ be a set of sequences over an arbitrary input domain $\Sigma_0$. Assume that some computational system $M$, affected by a piecewise uniformly continuous noise process $Z$, recognizes $L$ with reliability $\rho$, for some arbitrary $\rho > 0$. Then $L$ is regular.*

The *proof* of Theorem 3.1 relies on an analysis of the space of probability density functions over the state set $\Omega$. An upper bound on the number of states of a deterministic finite automaton that simulates $M$ can be given in terms of the number $k$ of components $\Omega_j$ of the state set $\Omega$, the dimension and diameter of $\Omega$, a bound on the values of the noise density function $z$, and the value of $\delta$ for $\varepsilon = \rho/4\mu(\Omega)$ in condition (1). For details we refer to [Maass, Orponen, 1996].[5]     ∎

**Remark 3.2** *In stark contrast to the results of [Siegelmann, Sontag, 1991] and [Maass, 1996] for the noise-free case, the preceding Theorem implies that both recurrent analog neural nets and recurrent networks of spiking neurons with online input from $\Sigma_0^*$ can only recognize regular languages in the presence of any reasonable type of analog noise, even if their computation time is unlimited and if they employ arbitrary real-valued parameters.*

Let us say that a noise process $Z$ defined on a set $\Omega \subseteq \mathbf{R}^d$ is *bounded by* $\eta$ if it can move a state $p$ only to other states $q$ that have a distance $\leq \eta$ from $p$ in the $L_1$-norm over $\mathbf{R}^d$ , i.e. if its density kernel $z$ has the property that for any $p = \langle p_1, \ldots, p_d \rangle$ and $q = \langle q_1, \ldots, q_d \rangle \in \Omega$, $z(p,q) > 0$ implies that $|q_i - p_i| \leq \eta$ for $i = 1, \ldots, d$. Obviously $\eta$-bounded noise processes are a very special class. However they provide an example which shows that the general upper bound of Theorem 3.1 is a sense optimal:

**Theorem 3.3** *For every regular language $L \subseteq \{-1, 1\}^*$ there is a constant $\eta > 0$ such that $L$ can be recognized with perfect reliability (i.e. $\rho = \frac{1}{2}$) by a recurrent analog neural net in spite of any noise process $Z$ bounded by $\eta$.* ∎

We now consider the effect of analog noise on discrete time analog computations with *batch-input*. The *proofs* of Theorems 3.4 and 3.5 are quite complex (see [Maass, Orponen, 1996]).

**Theorem 3.4** *There exists a finite upper bound for the VC-dimension of layered feedforward sigmoidal neural nets and feedforward networks of spiking neurons with piecewise uniformly continuous analog noise (for arbitrary real-valued inputs, Boolean output computed with some arbitrary reliability $\rho > 0$, and arbitrary real-valued "programmable parameters") which does not depend on the size or structure of the network beyond its first hidden layer.* ∎

**Theorem 3.5** *There exists a finite upper bound for the VC-dimension of recurrent sigmoidal neural nets and networks of spiking neurons with piecewise uniformly continuous analog noise (for arbitrary real valued inputs, Boolean output computed with some arbitrary reliability $\rho > 0$, and arbitrary real valued "programmable parameters") which does not depend on the computation time of the network, even if the computation time is allowed to vary for different inputs.* ∎

## 4    Conclusions

We have introduced a new framework for the analysis of analog noise in discrete-time analog computations that is better suited for "real-world" applications and

---

of recurrent neural nets with bounded noise and $\rho = 1/2$ , i.e. for certain computations with *perfect reliability*. This case may not require the consideration of probability density functions. However it turns out that the proof for this special case in [Casey, 1996] is wrong. The proof of Corollary 3.1 in [Casey, 1996] relies on the argument that a compact set "can contain only a finite number of disjoint sets with nonempty interior". This argument is wrong, as the counterexample of the intervals $[1/(2i + 1), 1/2i]$ for $i = 1, 2, \ldots$ shows. These infinitely many disjoint intervals are all contained in the compact set $[0, 1]$ . In addition, there is an independent problem with the *structure* of the proof of Corollary 3.1 in [Casey, 1996]. It is derived as a consequence of the proof of Theorem 3.1 in [Casey, 1996]. However that proof relies on the *assumption* that the recurrent neural net accepts a regular language. Hence the proof via probability density functions in [Maass, Orponen, 1996] provides the first valid proof for the claim of Corollary 3.1 in [Casey, 1996].

more flexible than previous models. In contrast to preceding models it also covers
important concrete cases such as analog neural nets with a Gaussian distribution of
noise on analog gate outputs, noisy computations with less than perfect reliability,
and computations in networks of noisy spiking neurons.

Furthermore we have introduced adequate mathematical tools for analyzing the
effect of analog noise in this new framework. These tools differ quite strongly from
those that have previously been used for the investigation of noisy computations.
We show that they provide new bounds for the computational power and VC-
dimension of analog neural nets and networks of spiking neurons in the presence of
analog noise.

Finally we would like to point out that our model for noisy analog computations
can also be applied to completely different types of models for discrete time analog
computation than neural nets, such as arithmetical circuits, the random access
machine (RAM) with analog inputs, the parallel random access machine (PRAM)
with analog inputs, various computational discrete-time dynamical systems and
(with some minor adjustments) also the BSS model [Blum, Shub, Smale, 1989]. Our
framework provides for each of these models an adequate definition of noise-robust
computation in the presence of analog noise, and our results provide upper bounds
for their computational power and VC-dimension in terms of characteristica of their
analog noise.

## Footnotes

* Klosterwiesgasse 32/2, A–8010 Graz, Austria. E-mail: maass@igi.tu-graz.ac.at.

† P. O. Box 35, FIN–40351 Jyväskylä, Finland. E-mail: orponen@math.jyu.fi. Part of this work was done while this author was at the University of Helsinki, and during visits to the Technische Universität Graz and the University of Chile in Santiago.

[1] There are relatively few examples for nontrivial computations on common digital or analog computational models that can achieve *perfect reliability* of the output in spite of noisy internal components. Most constructions of noise-robust computational models rely on the *replication* of noisy computational units (see [von Neumann, 1956], [Cowan, 1966]). The idea of this method is that the average of the outputs of $k$ identical noisy computational units (with stochastically independent noise processes) is with high probability close to the expected value of their output, if $k$ is sufficiently large. However for *any* value of $k$ there exists in general a small but nonzero probability that this average deviates strongly from its expected value. In addition, if one assumes that the computational unit that produces the output of the computations is also noisy, one cannot expect that the reliability of the output of the computation is larger than the reliability of this last computational unit. Consequently there exist many methods for *reducing* the error-probability of the output *to a small value*, but these methods cannot achieve error probability 0 at the output.

[2]For example, one might think that analog noise which is likely to move an internal state over a *large* distance is more harmful than another type of analog noise which keeps an internal state within its *neighborhood*. However this intuition is deceptive. Consider the extreme case of analog noise in a sigmoidal neural net which moves a gate output $x \in [-1, 1]$ to a value in the $\varepsilon$-neighborhood of $-x$. This type of noise moves some values $x$ over large distances, but it appears to be less harmful for noise-robust computing than noise which moves $x$ to an arbitrary value in the $10\varepsilon$-neighborhood of $x$.

[3]In this case one wants to set $\Omega_{sp} := (\bigcup_{j=1}^{l} [0, T)^j \cup \{not\text{-}firing\})^m$, where $T > 0$ is a sufficiently large constant so that it suffices to consider only the firing history of the network during a preceding time interval of length $T$ in order to determine whether a neuron fires (e.g. $T = 30$ ms for a biological neural system). If one partitions the time axis into discrete time windows $[0, T)$, $[T, 2T)$, ... , then in the noise-free case the firing events during each time window are completely determined by those in the preceding one. A component $p_i \in [0, T)^j$ of a state in this set $\Omega_{sp}$ indicates that the corresponding neuron $i$ has fired exactly $j$ times during the considered time interval, and it also specifies the $j$ firing times of this neuron during this interval. Due to refractory effects one can choose $l < \infty$ for biological neural systems, e.g. $l = 15$ for $T = 30$ ms. With some straightforward formal operations one can also write this state set $\Omega_{sp}$ as a bounded subset of $\mathbf{R}^d$ for $d := l \cdot m$.

[4]We would like to thank Peter Auer for helpful conversations on this topic.

[5]A corresponding result is claimed in Corollary 3.1 of [Casey, 1996] for the special case

# References

[Blum, Shub, Smale, 1989] L. Blum, M. Shub, S. Smale, On a theory of computation
over the real numbers: NP-completeness, recursive functions and universal machines.
*Bulletin of the Amer. Math. Soc. 21* (1989), 1–46.

[Casey, 1996] M. Casey, The dynamics of discrete-time computation, with application to
recurrent neural networks and finite state machine extraction. *Neural Computation 8*
(1996), 1135–1178.

[Cowan, 1966] J. D. Cowan, Synthesis of reliable automata from unreliable components.
*Automata Theory* (E. R. Caianiello, ed.), 131–145. Academic Press, New York, 1966.

[Maass, 1996] W. Maass, Lower bounds for the computational power of networks of spiking
neurons. *Neural Computation 8* (1996), 1–40.

[Maass, 1997] W. Maass, Fast sigmoidal networks via spiking neurons, to appear in
*Neural Computation 9*, 1997. FTP-host: archive.cis.ohio-state.edu, FTP-filename:
/pub/neuroprose /maass.sigmoidal-spiking.ps.Z.

[Maass, Orponen, 1996] W. Maass, P. Orponen, On the effect of analog noise in
discrete-time analog computations (journal version), submitted for publication; see
http://www.math.jyu.fi/~orponen/papers/noisyac.ps.

[Pippenger, 1989] N. Pippenger, Invariance of complexity measures for networks with un-
reliable gates. *J. Assoc. Comput. Mach. 36* (1989), 531–539.

[Rabin, 1963] M. Rabin, Probabilistic automata. *Information and Control 6* (1963), 230–
245.

[Siegelmann, 1994] H. T. Siegelmann, On the computational power of probabilistic and
faulty networks. *Proc. 21st International Colloquium on Automata, Languages, and
Programming*, 23–34. Lecture Notes in Computer Science 820, Springer-Verlag, Berlin,
1994.

[Siegelmann, Sontag, 1991] H. T. Siegelmann, E. D. Sontag, Turing computability with
neural nets. *Appl. Math. Letters 4(6)* (1991), 77–80.

[von Neumann, 1956] J. von Neumann, Probabilistic logics and the synthesis of reliable
organisms from unreliable components. *Automata Studies* (C. E. Shannon, J. E. Mc-
Carthy, eds.), 329–378. Annals of Mathematics Studies 34, Princeton University Press,
Princeton, NJ, 1956.